# Audio-Visual Sound Separation Via Hidden Markov Models

**John Hershey**
Department of Cognitive Science
University of California San Diego
jhershey@cogsci.ucsd.edu

Michael Casey
Mitsubishi Electric Research Labs
Cambridge, Massachussets
casey@merl.com

## Abstract

It is well known that under noisy conditions we can hear speech much more clearly when we read the speaker's lips. This suggests the utility of audio-visual information for the task of speech enhancement. We propose a method to exploit audio-visual cues to enable speech separation under non-stationary noise and with a single microphone. We revise and extend HMM-based speech enhancement techniques, in which signal and noise models are factorially combined, to incorporate visual lip information and employ novel signal HMMs in which the dynamics of narrow-band and wide band components are factorial. We avoid the combinatorial explosion in the factorial model by using a simple approximate inference technique to quickly estimate the clean signals in a mixture. We present a preliminary evaluation of this approach using a small-vocabulary audio-visual database, showing promising improvements in machine intelligibility for speech enhanced using audio and visual information.

## 1 Introduction

We often take for granted the ease with which we can carry on a conversation in the proverbial cocktail party scenario: guests chatter, glasses clink, music plays in the background: the room is filled with ambient sound. The vibrations from different sources and their reverberations coalesce translucently yielding a single time series at each ear, in which sounds largely overlap even in the frequency domain. Remarkably the human auditory system delivers high-quality impressions of sounds in conditions that perplex our best computational systems. A variety of strategies appear to be at work in this, including binaural spatial analysis, and inference using prior knowledge of likely signals and their contexts. In speech perception, vision often plays a crucial role, because we can follow in the lips and face the very mechanisms that modulate the sound, even when the sound is obscured by acoustic noise.

It has been demonstrated that the addition of visual cues can enhance speech recognition as much as removing 15 dB of noise [1]. Vision provides speech cues that are complementary to audio cues such as components of consonants and vowels that are likely to be obscured by acoustic noise [2]. Visual information is demonstra-

bly beneficial to HMM-based automatic speech recognition (ASR) systems, which typically suffer tremendously under moderate acoustical noise [3].

We introduce a method of audio-visual speech enhancement using factorial hidden Markov models (fHMMs). We focus on speech enhancement rather than speech recognition for two reasons: first, speech conveys useful paralinguistic information, such as prosody, emotion, and speaker identity, and second, speech contains useful cues for separation from noise, such as pitch. In automatic speech recognition (ASR) systems, these cues are typically discarded in an effort to reduce irrelevant variance among speakers and utterances within a phonetic class.

Whereas the benefit of vision to speech recognition is well known, we may well wonder if visual input offers similar benefits to speech enhancement. In [4] a non-parametric density estimator was used to adapt audio and video transforms to maximize the mutual information between the face of a target speaker and an audio mixture containing both the target voice and a distracter voice. These transforms were then used to construct a stationary filter for separating the target voice from the mixture without any prior knowledge or training. In [5] a multi-layer perceptron is trained to map noisy estimates of formants to clean ones, employing lip parameters (width, height and area of the lip opening) extracted from video as additional input. The re-estimated formant contours were used to filter the speech to enhance the signal. In both cases video information improved signal separation. Neither system, however, made use of the dynamics of speech.

In speech recognition, HMMs are commonly used because of the advantages of modeling signal dynamics. This suggests the following strategy: train an audio-visual HMM on clean speech, infer the likelihoods of its state sequences, and use the inferred state probabilities of the signal and noise to estimate a sequence of filters to clean the data. In cases where background noise also has regularity, such as the combination of two voices, another HMM can be used to model the background noise. Ephraim [6] first proposed an approach to factorially combining two HMMs in such an enhancement system. In [7] an efficient variational learning rule for the factorial HMM is formulated, and in [8, 9] fHMM speech enhancement was recently revived using some clever tricks to allow more complex models.

The fHMM approach is amenable to audio-visual speech enhancement in many different forms. In the simplest formulation, which we pursue here, the signal observation model includes visual features. These visual inputs constrain the signal HMM and produce more accurate filters. Below we present a prototype architecture for such a system along with preliminary results.[1]

## 1.1 Factorial Speech Models

One of the challenges of using speech HMMs for enhancement is to model speech in sufficient detail. Typically, speech models, following the practice in ASR, ignore *narrow-band*, spectral details (corresponding to upper cepstral components) which carry pitch information, because they tend to vary across speakers and utterances for the same word or phoneme. Instead such systems focus on the smooth, or *wide-band*, spectral characteristics (corresponding to lower cepstral components) such as are produced by the articulation of the mouth. Such wide-band spectral patterns loosely represent *formant* patterns, a well-known cue for vowel discrimination. In cases where the pitch or other narrow-band properties, of the background signals differ from the foreground speech, and have predictable dynamics, such as with

two simultaneous speech signals, these components may be helpful in separating the two signals. Figure 1 illustrates the analysis of two words into wide-band and narrow-band components.

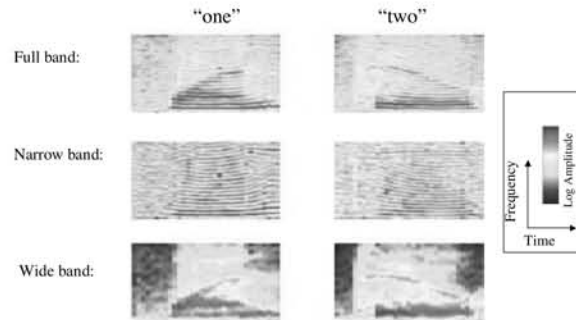

Figure 1: full-band, narrow-band, and wide-band log spectrograms of two words. The wide-band log spectrograms (bottom) are derived by low-pass filtering the log spectra (across the frequency domain), and the narrow-band log spectrograms (middle) derived by high pass filtering the log spectra The full log spectrogram (top) is the sum of the two.

However, the wide-band and narrow-band variations in speech are only loosely coupled. For instance, a given formant is likely to be uttered with many different pitches and a given pitch may be used to utter any formant. Thus a model of the full spectrum of speech would have to have enough states to represent every combination of pitches and formants. Such a model requires a large amount of training data and imposes serious computational burdens. For instance in [8] a model with 8000 states is employed. When combined with a similarly complex noise model, the composite model has 64 million states. This is expensive in terms of computation as well as the number of data points required for inference.

To parsimoniously model the complexity of speech, we employ a factorial HMM for a single speech signal, in which wide and narrow-band components are represented in sub-models with independent dynamics. We therefore train the two submodels independently using Gaussian observation probability density functions (p.d.f.) on the wide-band or narrow-band log spectra, with diagonal covariances for the sake of simplicity. Figure 2(a) depicts the graphical model for a single wide or narrow-band component.

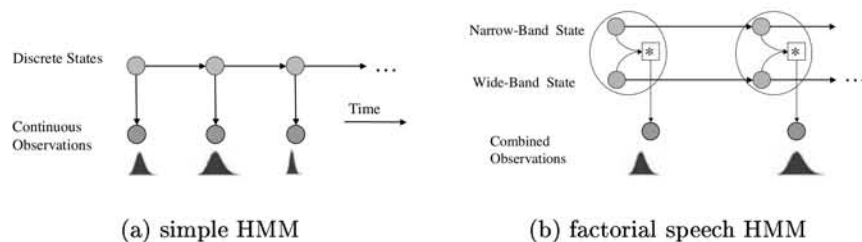

(a) simple HMM          (b) factorial speech HMM

Figure 2: single HMMs are trained separately on wide-band and narrow-band speech signals (a) and then combined factorially in (b) by adding the means and variances of their observation distributions

To combine the sub-models, we have to specify the observation p.d.f. for a combination of a wide and a narrow-band state, over the log-spectrum of speech prior to liftering. Because the observation densities of each component are Gaussian, and the log-spectra of the wide and narrow-band components add in the log spectrum, the composite state has a Gaussian observation p.d.f., whose mean and variance is the sum of the component observation means and variances. Although the states of the two sub-models are marginally independent they are typically conditionally dependent given the observation sequence. In other words we assume that the state dependencies between the sub-models for a given speech signal can be explained entirely via the observations. Figure 2(b) depicts the combination of the wide and narrow-band models, where the observation p.d.f.'s are a function of two state variables.

When combining the signal and noise models (or two different speech models) in contrast, the signals add in the frequency domain, and hence in the log spectral domain they longer simply add. In the spectral domain the amplitudes of the two signals have log-normal distributions, and the relative phases are unknown. There is no closed form distribution for the sum of two random variables with log-normal amplitudes and a uniformly distributed phase difference. Disregarding phase differences we apply a well-known approximation to the sum of two lognormal random variables, in which we match the mean and variance of a lognormal random variable to the sum of the means and variances of the two component lognormal random variables [10]. Phase uncertainty can also be incorporated into an approximation; however in practice the costs appear to outweigh the benefits.[2] Figure 3(a) depicts the combination of two factorial speech models, where the observation p.d.f.s are a function of two state variables.

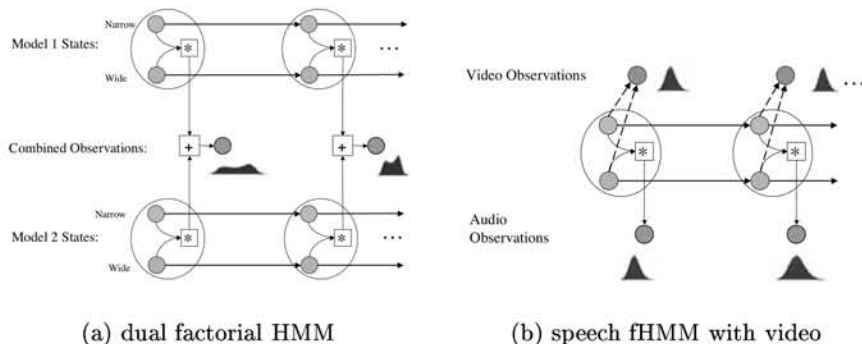

(a) dual factorial HMM        (b) speech fHMM with video

Figure 3: combining two speech fHMMs (a) and adding video observations to a speech fHMM (b).

Using the log-normal observation distribution of the composite model we can estimate the likelihood of the speech and noise states for each frame using the well known forward-backward recursion. For each frame of the test data we can compute the expected value of the amplitude of each model in each frequency bin. Taking

the expected value of the signal in the numerator and the expected value of the signal plus noise in the denominator yields a Wiener filter which is applied to the original noisy signal enhancing the desired component. When we have two speech signals one person's noise is another's signal and we can separate both by the same method.

## 2 Incorporating vision

We incorporate vision after training the audio models in order to test the improvement yielded by visual input while holding the audio model constant. A video observation distribution is added to each state in the model by obtaining the probability of each state in each frame of the audio training data using the forward-backward procedure, then estimating the parameters of the video observation distributions for each state, in the manner of the Baum-Welch observation re-estimation formula. This procedure is iterated until it converges. In this way we construct a system in which the visual observations are modular. Figure 3(b) depicts the structure of the resulting speech model.

Such a method in which audio and visual features are integrated early in processing is only one of several approaches. We envision other late integration approaches in which audio and visual dynamics are more loosely coupled. What method of audio-visual integration may be best for this task is an open question.

## 3 Efficient inference

In the models described above, in which we factorially combine two speech models, each of which is itself factorial, the complexity of inference in the composite model, using the forward-backward recursion, can easily become unmanageable. If $K$ is the number of states in each subcomponent, then $K^4$ is the number of states in the composite HMM. In our experiments $K$ is on the order of 40 states, so there are 2,560,000 states in the composite model. Naively each composite state must be searched when computing the probabilities of state sequences necessary for inference. Interesting approximation schemes for similar models are developed in [8, 9]. We develop an approximation as follows.

Rather than computing the forward-backward procedure on the composite HMM, we compute it sequentially on each sub-HMM to derive the probability of each state in each frame. Of course, in order to evaluate the observation probabilities of the current sub-HMMs for a given frame, we need to consider the state probabilities of the other three sub-HMMs, because their means and variances are combined in the observation model. These state probabilities and their associated observation probabilities comprise a mixture model for a given frame. The composite mixture model still has $K^4$ states, so to defray this complexity during forward-backward analysis of the current sub-HMM, for each frame we approximate the observation mixtures of each of the other three sub-HMMs with a single Gaussian, whose mean and variance matches that of the mixture. Thus we only have to consider the $K$ states of the current model, and use the summarized means and variances of the other three HMMs as auxiliary inputs to the observation model. We initialize the state probabilities in each frame with the equilibrium distribution for each sub-HMM. In our experiments, after a handful of iterations, the composite state probabilities tend to converge. This method is closely related to a structured variational approximation for factorial HMMs [7] and can be also be seen as an approximate belief propagation or sum-product algorithm [11].

## 4 Data

We used a small-vocabulary audio-visual speech database developed by Fu Jie Huang at Carnegie Mellon University[3] [12]. These data consist of audio and video recordings of 10 subjects (7 males and 3 females) saying 78 isolated words commonly used for numbers and time, such as,"one" "Monday", "February", "night", etc. The sequence of 78 words is repeated in 10 different takes. Half of these takes were used for training, and one of the remaining takes was used as the test set.

The data set included outer lip parameters extracted from video using an automatic lip tracker, including height of the upper and lower lips relative to the corners the width from corner to corner. We interpolated these lip parameters to match the audio frame rate, and calculate time derivatives.

Audio consisted of 16-bit, 44.1 kHz recordings which we resample to 8000 kHz. The audio was framed at 60 frames per second, with an overlap of 50%, yielding 264 samples per frame.[4] The frames were analyzed into cepstra: the wide-band log spectrum is derived from the lower 20 cepstral components and the wide-band log spectrum from the upper cepstra.

## 5 Results

Speaker dependent wide and narrow-band HMMs having 40 states each were trained on data from two subjects ("Anne" and "Chris") selected from the training set. A PCA basis was used to reduce the log spectrograms to a more manageable size of 30 dimensions during training. This resulted in some non-zero covariances near the diagonal in the learned observation covariance matrices, which we discarded. An entropic prior and parameter extinction were used to sparsify the transition matrices during training [13].

The narrow-band model learned states that represented different pitches and had transition probabilities that were non-zero mainly between neighboring pitches. The narrow-band model's video observation probability distributions were largely overlapping, reflecting the fact that video tells us little about pitch. The wide-band model learned states that represented different formant structures. The video observation distributions for several states in the wide-band model were clearly separated, reflecting the information that video provides about the formant structure.

Subjectively the enhanced signals sound well separated from each other for the most part. Figure 4(a) (bottom) shows the estimated spectrograms for a mixture of two different words spoken by the same speaker – an extremely difficult task. To quantify these results we evaluate the system using speech recognizer, on the slightly easier task of separating the speech of the two different speakers, whose voices were in different but overlapping pitch ranges.

A test set was generated by mixing together 39 randomly chosen pairs of words, one from each subject, such that no word was used twice. Each word pair was mixed at five different signal to noise ratios (SNRs), with the SNR provided to the system at test time.[5] The total number of test mixtures for each subject was thus 195.

The separated test sounds were estimated by the system under two conditions: with and without the use of video information.

We evaluated the estimates on the test set using a speech recognition system developed by Bhiksha Raj, using the CMU Sphinx ASR engine.[6] Existing speech HMMs trained on 60 hours of broadcast news data were used for recognition.[7] The models were adapted in an unsupervised manner to clean speech from each speaker, by learning a single affine transformation of all the state means, using a maximum likelihood linear regression procedure [14]. The recognizer adapted to each speaker was tested with the enhanced speech produced by the speech model for that speaker, as well as with no enhancement.

Results are shown in figure 4(b). Recognition was greatly facilitated by the enhancement, with additional gains resulting from the use of video. It is somewhat surprising that the gains for video occur mostly in areas of higher SNR, whereas in human speech perception they occur under lower SNR. Little subjective difference was noted with the use of video in the case of two speakers. However in other experiments, when both voices came from the same speaker, the video was crucial in disambiguating which signal came from which voice.

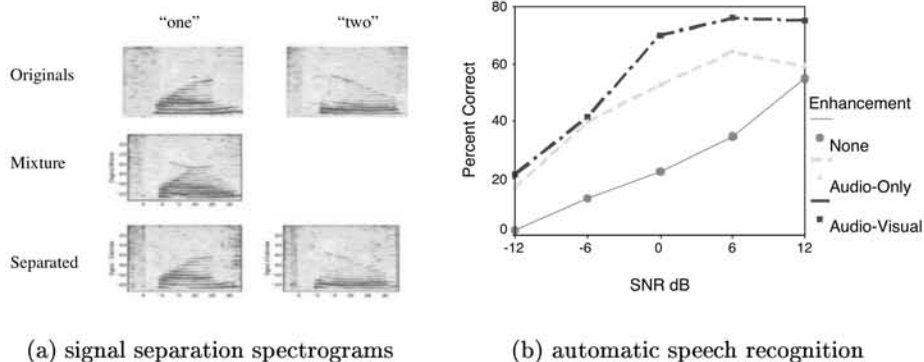

(a) signal separation spectrograms          (b) automatic speech recognition

Figure 4: spectrograms of separated speech signals for a mixture two words spoken by the same speaker (a), and speech recognition performance for 39 mixtures of two words spoken by different speakers (b)

# 6    Discussion

We have presented promising techniques for audio-visual speech enhancement. We introduced a factorial HMM to track both formant and pitch information, as well as video, in a unified probabilistic model, and demonstrated its effectiveness in speech enhancement. We are not aware of any other HMM-based audio-visual

---

elsewhere [6] and is beyond the scope of this paper.

[6]see http://www.speech.cs.cmu.edu/sphinx/.

[7]These models represented every combination of three phones (*triphones*) using 6000 states tied across triphone models, with a 16-element Gaussian mixture observation model for each state. The data were processed at 8 kHz in 25ms windows overlapped by 15ms, with a frame rate of 100 frames per second, and analyzed into 31 Mel frequency components from which 13 cepstral coefficients were derived. These coefficients with the mean vector removed, and supplemented with their time differences, comprised the observed features

speech enhancement systems in the literature. The results are tentative given the small sample of voices used; however they suggest that further study with a larger sample of voices is warranted. It would be useful to compare the performance of a factorial speech model to that of each factor in isolation, as well as to a full-spectrum model. Measures of quality and intelligibility by human listeners in terms of speech and emotion recognition, as well as speaker identity, will also be helpful in further demonstrating the utility of these techniques. We look forward to further development of these techniques in future research.

**Acknowledgments**

We wish to thank Mitsubishi Electric Research Labs for hosting this research. Special thanks to Bhiksha Raj for devising and producing the evaluation using speech recognition, and to Matt Brand for his entropic HMM toolkit.

## Footnotes

[1]We defer a detailed mathematical development to subsequent publications. Contact jhershey@cogsci.ucsd.edu for further information

[2]The uncertainty of the phase differences can be incorporated by modeling the sum as a mixture of lognormals that uniformly samples phase differences. Each mixture element is approximated by taking as its mean the length of the sum of the mean amplitudes when added in the complex plane according a particular phase difference, and as its variance the sum of the two variances. This estimation is facilitated by the assumption of diagonal covariances in the log spectral domain.

[3]see http://amp.ece.cmu.edu/projects/AudioVisualSpeechProcessing/

[4]Sine windows were used in analysis and synthesis such that their product forms windows that sum to unity when overlapped 50%. The windowed frames were analyzed using a 264-point fast Fourier transform (FFT). The phases of the resulting spectra were discarded.

[5]Estimation of the SNR is necessary in practice; however this subject has been treated

# References

[1] W. H. Sumby and I. Pollack. Visual contribution to speech intelligibility in noise. *Journal of the Acoustical Society of America*, 26:212–215, 1954.

[2] Jordi Robert-Ribes, Jean-Luc Schwartz, Tahar Lallouache, and Pierre Escudier. Complementarity and synergy in bimodal speech. *Journel of the Acoustical Society of America*, 103(6):3677–3689, 1998.

[3] Stepmane Dupont and Juergen Luettin. Audio-visual speech modeling for continuous speech recognition. *IEEE transactions on Multimedia*, 2(3):141–151, 2000.

[4] John W. Fisher, Trevor Darrell, William T. Freeman, and Paul Viola. Learning joint statistical models for audio-visual fusion and segregation. In *Advances in Neural Information Processing Systems 13*. 2001.

[5] Laurent Girin, Jean-Luc Schwartz, and Gang Feng. Audio-visual enhancement of speech in noise. *Journel of the Acoustical Society of America*, 109(6):3007–3019, 2001.

[6] Yariv Ephraim. Statistical-model based speech enhancement systems. *Proceedings of the IEEE*, 80(10):1526–1554, 1992.

[7] Z. Ghahramani and M. Jordan. Factorial hidden markov models. In David S. Touretzky, Michael C. Mozer, and M.E. Hasselmo, editors, *Advances in Neural Information Processing Systems 8*, 1996.

[8] Sam T. Roweis. One microphone source separation. In *Advances in Neural Information Processing Systems 13*. 2001.

[9] Hagai Attias, John C. Platt, Alex Acero, and Li Deng. Speech denoising and dereverberation using probabilistic models. In *Advances in Neural Information Processing Systems 13*. 2001.

[10] M. J. F. Gales. *Model-Based Techniques for Noise Robust Speech Recognition*. PhD thesis, Cambridge University, 1996.

[11] F. R. Kschischang, B. Frey, and H.-A. Loeliger. Factor graphs and the sum-product algorithm. *IEEE Trans. Inform. Theory*, 47(2):498–519, 2001.

[12] F. J. Huang and T. Chen. Real-time lip-synch face animation driven by human voice. In *IEEE Workshop on Multimedia Signal Processing*, Los Angeles, California, Dec 1998.

[13] Matt Brand. Structure learning in conditional probability models via an entropic prior and parameter extinction. *Neural Computation*, 11(5):1155–1182, 1999.

[14] C. J. Leggetter and P. C. Woodland. Maximum likelihood linear regression for speaker adaptation of the parameters of continuous density hidden markov models. *Computer Speech and Language*, 9:171–185, 1995.
